# MATHEMATICAL ANALYSIS OF LEARNING BEHAVIOR
## OF NEURONAL MODELS

BY
JOHN Y. CHEUNG
MASSOUD OMIDVAR

SCHOOL OF ELECTRICAL ENGINEERING AND COMPUTER SCIENCE
UNIVERSITY OF OKLAHOMA
NORMAN, OK 73019

Presented to the IEEE Conference on "Neural Information Processing Systems–
Natural and Synthetic," Denver, November 8–12, 1987, and to be published in
the Collection of Papers from the IEEE Conference on NIPS.

Please address all further correspondence to:

John Y. Cheung
School of EECS
202 W. Boyd, CEC 219
Norman, OK 73019
(405)325-4721

November, 1987

# MATHEMATICAL ANALYSIS OF LEARNING BEHAVIOR OF NEURONAL MODELS

John Y. Cheung and Massoud Omidvar
School of Electrical Engineering
and Computer Science

## ABSTRACT

In this paper, we wish to analyze the convergence behavior of a number of neuronal plasticity models. Recent neurophysiological research suggests that the neuronal behavior is adaptive. In particular, memory stored within a neuron is associated with the synaptic weights which are varied or adjusted to achieve learning. A number of adaptive neuronal models have been proposed in the literature. Three specific models will be analyzed in this paper, specifically the Hebb model, the Sutton–Barto model, and the most recent trace model. In this paper we will examine the conditions for convergence, the position of convergence and the rate at convergence, of these models as they applied to classical conditioning. Simulation results are also presented to verify the analysis.

## INTRODUCTION

A number of static models to describe the behavior of a neuron have been in use in the past decades. More recently, research in neurophysiology suggests that a static view may be insufficient. Rather, the parameters within a neuron tend to vary with past history to achieve learning. It was suggested that by altering the internal parameters, neurons may adapt themselves to repetitive input stimuli and become conditioned. Learning thus occurs when the neurons are conditioned. To describe this behavior of neuronal plasticity, a number of models have been proposed. The earliest one may have been postulated by Hebb and more recently by Sutton and Barto [1]. We will also introduce a new model, the most recent trace (or MRT) model in this paper. The primary objective of this paper, however, is to analyze the convergence behavior of these models during adaptation.

The general neuronal model used in this paper is shown in Figure 1. There are a number of neuronal inputs $x_i(t), i = 1, \ldots, N$. Each input is scaled by the corresponding synaptic weights $w_i(t), i = 1, \ldots, N$. The weighted inputs are arithmetically summed.

$$y(t) = \sum_{i=1}^{N} x_i(t) w_i(t) - \Theta(t) \tag{1}$$

where $\Theta(t)$ is taken to be zero.

Neuronal inputs are assumed to take on numerical values ranging from zero to one inclusively. Synaptic weights are allowed to take on any reasonable values for the purpose of this paper though in reality, the weights may very well be bounded. Since the relative magnitude of the weights and the neuronal inputs are not well defined at this point, we will not put a bound on the magnitude of the weights also. The neuronal output is normally the result of a sigmoidal transformation. For simplicity, we will approximate this operation by a linear transformation.

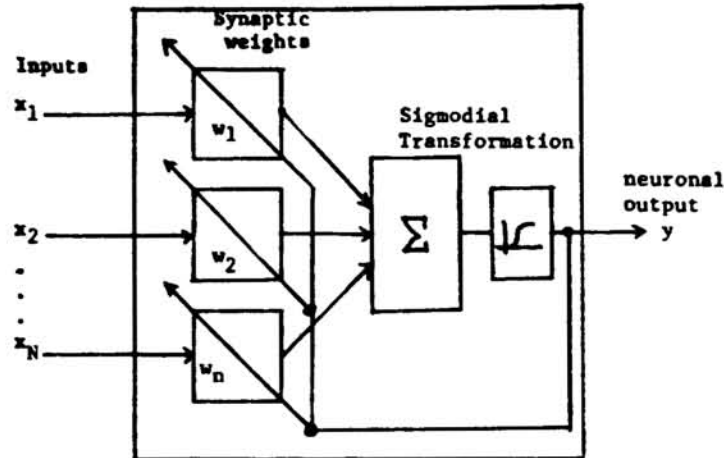

Figure 1.  A general neuronal model.

For convergence analysis, we will assume that there are only two neuronal inputs in the traditional classical conditioning environment for simplicity. Of course, the analysis techniques can be extended to any number of inputs. In classical conditioning, the two inputs are the conditioned stimulus $x_c(t)$ and the unconditioned stimulus $x_u(t)$.

## THE SUTTON–BARTO MODEL

More recently, Sutton and Barto [1] have proposed an adaptive model based on both the signal trace $\bar{x}_i(t)$ and the output trace $\bar{y}(t)$ as given below:

$$w_i(t+1) = w_i(t) + c\bar{x}_i(t)(y(t)) - \bar{y}(t) \tag{2a}$$

$$\bar{y}(t+1) = \beta\bar{y}(t) + (1-\beta)y(t) \tag{2b}$$

$$\bar{x}_i(t+1) = \alpha\bar{x}_i(t) + x_i(t) \tag{2c}$$

where both $\alpha$ and $\beta$ are positive constants.

## Condition of Convergence

In order to simplify the analysis, we will choose $\alpha = 0$ and $\beta = 0$, i.e.:

$$\bar{x}_i(t) = x_i(t-1)$$

and

$$\bar{y}(t) = y(t-1)$$

In other words, (2a) becomes:

$$w_i(t+1) = w_i(t) + cx_i(t)(y(t) - y(t-1)) \tag{3}$$

The above assumption only serves to simplify the analysis and will not affect the convergence conditions because the boundedness of $\bar{x}_i(t)$ and $\bar{y}(t)$ only depends on that for $x_i(t)$ and $y(t-1)$ respectively.

As in the previous section, we recognize that (3) is a recurrence relation so convergence can be checked by the ratio test. It is also possible to rewrite (3) in matrix format. Due to the recursion of the neuronal output in the equation, we will include the neuronal output $y(t)$ in the parameter vector also:

$$\begin{pmatrix} w_1(t+1) \\ w_2(t+1) \\ y(t) \end{pmatrix} = \begin{pmatrix} 1 + cx_1^2(t) & cx_1(t)x_2(t) & -cx_1(t) \\ cx_1(t)x_2(t) & 1 + cx_2^2(t) & -cx_2(t) \\ x_1(t) & x_2(t) & 0 \end{pmatrix} \begin{pmatrix} w_1(t) \\ w_2(t) \\ y(t-1) \end{pmatrix} \tag{4}$$

or

$$W^{(S-B)}(t+1) = A^{(S-B)} W^{(S-B)}(t)$$

To show convergence, we need to set the magnitude of the determinant of $A^{(S-B)}$ to be less than unity.

$$|A^{(S-B)}| = c(x_1^2(t) + x_2^2(t)) \tag{5}$$

Hence, the condition for convergence is:

$$c < \frac{1}{x_1^2(t) + x_2^2(t)} \tag{6}$$

From (6), we can see that the adaptation constant must be chosen to be less than the reciprocal of the Euclidean sum of energies of all the inputs. The same techniques can be extended to any number of inputs. This can be proved merely by following the same procedures outlined above.

## Position At Convergence

Having proved convergence of the Sutton–Barto model equations of neuronal plasticity, we want to find out next at what location the system remains when converged. We have seen earlier that at convergence, the weights cease to change and so does the neuronal output. We will denote this converged position as $(W^{(S-B)})^* \equiv W^{(S-B)}(\infty)$. In other words:

$$(W^{(S-B)})^* = A^{(S-B)}(W^{(S-B)})^* \tag{7}$$

Since any arbitrary parameter vector can always be decomposed into a weighted sum of the eigenvectors, i.e.

$$W^{(S-B)}(0) = \alpha_1 V_1 + \alpha_2 V_2 + \alpha_3 V_3 \tag{8}$$

The constants $\alpha_1$, $\alpha_2$, and $\alpha_3$ can easily be found by inverting $A^{(S-B)}$. The eigenvalues of $A^{(S-B)}$ can be shown to be 1, 1, and $c(x_1^2 + x_2^2)$. When $c$ is within the region of convergence, the magnitude of the third eigenvalue is less than unity. That means that at convergence, there will be no contribution from the third eigenvector. Hence,

$$(W^{(S-B)})^* = \lim_{t \to \infty} W^{(S-B)}(t) = \alpha_1 V_1 + \alpha_2 V_2 \tag{9}$$

From (9), we can predict precisely what the converged position would be given only with the initial conditions.

## Rate of Convergence

We have seen that when $c$ is carefully chosen, the Sutton–Barto model will converge and we have also derived an expression for the converged position. Next we want to find out how fast convergence can be attained. The rate of convergence is a measure of how fast the initial parameter approaches the optimal position. The asymptotic rate of convergence is[2]:

$$R_\infty(A^{(S-B)}) = -\log S(A^{(S-B)}) \tag{10}$$

where $S(A^{(S-B)})$ is the spectral radius and is equalled to $c(x_1^2 + x_2^2)$ in this case. This completes the convergence analysis on the Sutton–Barto model of neuronal plasticity.

## THE MRT MODEL OF NEURONAL PLASTICITY

The most recent trace (MRT) model of neuronal plasticity [3] developed by the authors can be considered as a cross between the Sutton–Barto model and the Klopf's model [4]. The adaptation of the synaptic weights can be expressed as follows:

$$w_i(t+1) = w_i(t) + cw_i(t)x_i(t)(y(t) - y(t-1)) \tag{11}$$

A comparison of (11) and the Sutton-Barto model in (3) shows that the second term on the right hand side contains an extra factor, $w_i(t)$, which is used to speed up the convergence as shown later. The output trace has been replaced by $y(t-1)$, the most recent output, hence the name, the most recent trace model. The input trace is also replaced by the most recent input.

## Condition of Convergence

We can now proceed to analyze the condition of convergence for the MRT model. Due to the presence of the $w_i(t)$ factor in the second term in (31), the ratio test cannot be applied here. To analyze the convergence behavior further, let us rewrite (11) in matrix format:

$$
\begin{pmatrix} w_1(t+1) \\ w_2(t+1) \\ y(t) \end{pmatrix} = \begin{pmatrix} 1 & 0 & 0 \\ 0 & 1 & 0 \\ x_1(t) & x_2(t) & 0 \end{pmatrix} \begin{pmatrix} w_1(t) \\ w_2(t) \\ y(t-1) \end{pmatrix}
$$

$$
+ c(w_1(t)\ w_2(t)\ y(t-1)) \begin{pmatrix} x_1(t) \\ x_2(t) \\ -1 \end{pmatrix} \begin{pmatrix} x_1(t) & 0 & 0 \\ 0 & x_2(t) & 0 \\ 0 & 0 & 0 \end{pmatrix} \begin{pmatrix} w_1(t) \\ w_2(t) \\ y(t-1) \end{pmatrix} \quad (12)
$$

or

$$
W^{(MRT)}(t+1) = A^{(MRT)} W^{(MRT)}(t) + c\ (W^{(MRT)}(t))^T BC W^{(MRT)}(t)
$$

The superscript $T$ denotes the matrix transpose operation. The above equation is quadratic in $W^{(MRT)}(t)$. Complete convergence analysis of this equation is extremely difficult.

In order to understand the convergence behavior of (12), we note that the dominant term that determines convergence mainly relates to the second quadratic term. Hence for convergence analysis only, we will ignore the first term:

$$
W^{(MRT)}(t+1) \approx c(W^{(MRT)}(t))^T BC W^{(MRT)}(t) \quad (13)
$$

We can readily see from above that the primary convergence factor is $B^T C$. Since $C$ is only dependent on $x_i(t)$, convergence can be obtained if the duration of the synaptic inputs being active is bounded. It can be shown that the condition of convergence is bounded by:

$$
c < \frac{1}{(x_1^2 w_1(\infty) + x_2^2 w_2(\infty))} \quad (14)
$$

We can readily see that the adaptation constant $c$ can be chosen according to (14) to ensure convergence for $t < T$.

## SIMULATIONS

To verify the theoretical analysis of these three adaptive neuronal models based on classical conditioning, these models have been simulated on the IBM 3081 mainframe using the FORTRAN language in single precision. Several test scenarios have been designed to compare the analytical predictions with actual simulation results.

To verify the conditions for convergence, we will vary the value of the adaptation constant $c$. The conditioned and unconditioned stimuli were set to unity and the value of $c$ varies between 0.1 to 1.0. For the Sutton–Barto model the simulation given in Fig. 2 shows that convergence is obtained for $c < 0.5$ as expected from theoretical analysis. For the MRT model, simulation results given in Fig. 3 shows that convergence is obtained for $c <$ 0.7, also as expected from theoretical analysis. The theoretical location at convergence for the Sutton and Barto model is also shown in Figure 2. It is readily seen that the simulation results confirm the theoretical expectations.

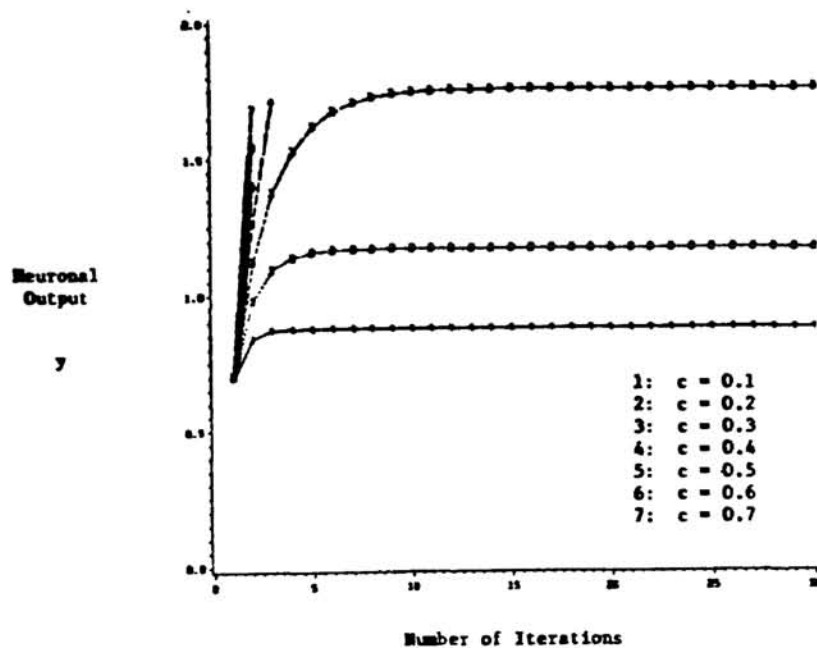

Figure 2. Plots of neuronal outputs versus the number of iterations for the Sutton–Barto model with different values of adaptation constant c.

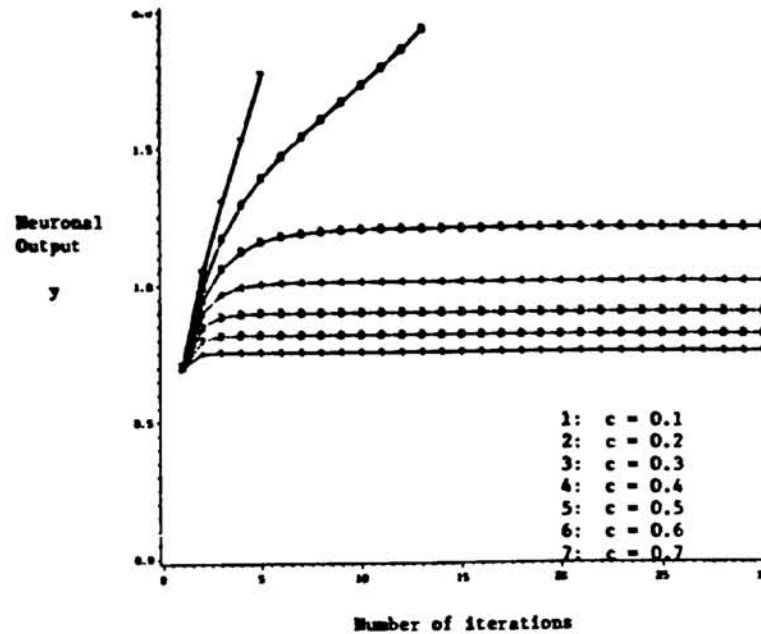

Figure 3. Plots of neuronal outputs versus the number of iterations for the MRT model with different values of adaptation constant c.

To illustrate the rate of convergence, we will plot the trajectory of the deviation in synaptic weights from the optimal values in the logarithmic scale since this error is logarithmic as found earlier. The slope of the line yields the rate of convergence. The trajectory for the Sutton–Barto Model is given in Figure 4 while that for the MRT model is given in Figure 5. It is clear from Figure 4 that the trajectory in the logarithmic form is a straight line. The slope $\hat{R}_n(A^{(S-B)})$ can readily be calculated. The curve for the MRT model given in Figure 5 is also a straight line but with a much larger slope showing faster convergence.

## SUMMARY

In this paper, we have sought to discover analytically the convergence behavior of three adaptive neuronal models. From the analysis, we see that the Hebb model does not converge at all. With constant active inputs, the output will grow exponentially. In spite of this lack of convergence the Hebb model is still a workable model realizing that the divergent behavior would be curtailed by the sigmoidal transformation to yield realistic outputs. The

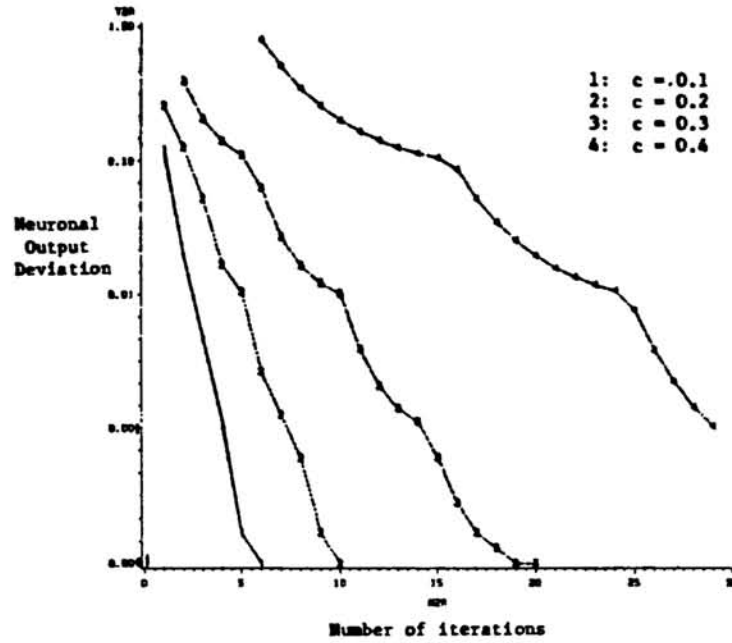

Figure 4. Trajectories of neuronal output deviations from static values for the Sutton-Barto model with different values of adaptation constant c.

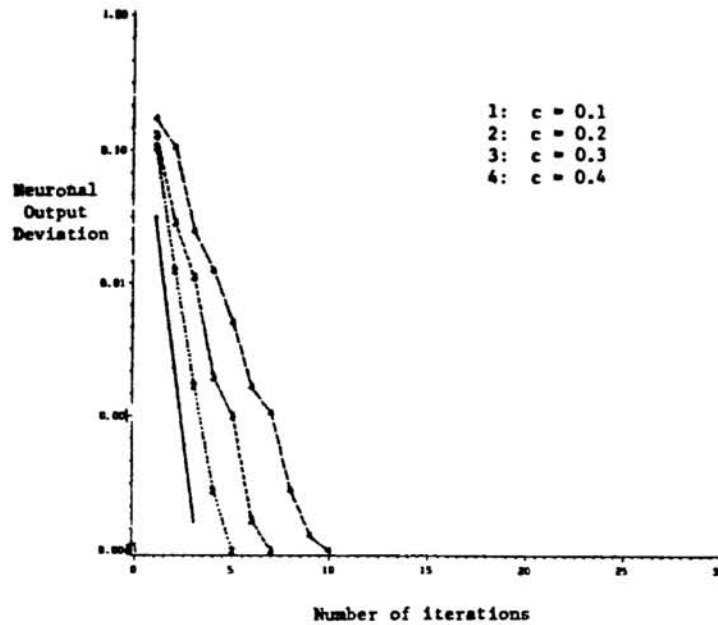

Figure 5. Trajectories of neuronal output deviations from static values for the MRT model with different values of adaptation constant c.

analysis on the Sutton and Barto model shows that this model will converge when the adaptation constant $c$ is carefully chosen. The bounds for $c$ is also found for this model. Due to the structure of this model, both the location at convergence and the rate of convergence are also found. We have also introduced a new model of neuronal plasticity called the most recent trace (MRT) model. Certain similarities exist between the MRT model and the Sutton–Barto model and also between the MRT model and the Klopf model. Analysis shows that the update equations for the synaptic weights are quadratic resulting in polynomial rate of convergence. Simulation results also show that much faster convergence rate can be obtained with the MRT model.

## REFERENCES

1. Sutton, R.S. and A.G. Barto, Psychological Review, vol. 88, p. 135, (1981).
2. Hageman, L. A. and D.M. Young. Applied Interactive Methods. (Academic Press, Inc. 1981).
3. Omidvar, Massoud. Analysis of Neuronal Plasticity. Doctoral dissertation, School of Electrical Engineering and Computer Science, University of Oklahoma, 1987.
4. Klopf, A.H. Proceedings of the American Institute of Physics Conference #151 on Neural Networks for Computing, p. 265–270, (1986).
